# Top-Down Control of Visual Attention:
# A Rational Account

**Michael C. Mozer**
*Dept. of Comp. Science &
Institute of Cog. Science
University of Colorado
Boulder, CO 80309 USA*

**Michael Shettel**
*Dept. of Comp. Science &
Institute of Cog. Science
University of Colorado
Boulder, CO 80309 USA*

**Shaun Vecera**
*Dept. of Psychology
University of Iowa
Iowa City, IA 52242 USA*

## Abstract

Theories of visual attention commonly posit that early parallel processes extract conspicuous features such as color contrast and motion from the visual field. These features are then combined into a saliency map, and attention is directed to the most salient regions first. Top-down attentional control is achieved by modulating the contribution of different feature types to the saliency map. A key source of data concerning attentional control comes from behavioral studies in which the effect of recent experience is examined as individuals repeatedly perform a perceptual discrimination task (e.g., "what shape is the odd-colored object?"). The robust finding is that repetition of features of recent trials (e.g., target color) facilitates performance. We view this facilitation as an adaptation to the statistical structure of the environment. We propose a probabilistic model of the environment that is updated after each trial. Under the assumption that attentional control operates so as to make performance more efficient for more likely environmental states, we obtain parsimonious explanations for data from four different experiments. Further, our model provides a rational explanation for why the influence of past experience on attentional control is short lived.

## 1 INTRODUCTION

The brain does not have the computational capacity to fully process the massive quantity of information provided by the eyes. Selective attention operates to filter the spatiotemporal stream to a manageable quantity. Key to understanding the nature of attention is discovering the algorithm governing selection, i.e., understanding what information will be selected and what will be suppressed. Selection is influenced by attributes of the spatiotemporal stream, often referred to as *bottom-up* contributions to attention. For example, attention is drawn to abrupt onsets, motion, and regions of high contrast in brightness and color. Most theories of attention posit that some visual information processing is performed preattentively and in parallel across the visual field. This processing extracts *primitive* visual features such as color and motion, which provide the bottom-up cues for attentional guidance. However, attention is not driven willy nilly by these cues. The deployment of attention can be modulated by task instructions, current goals, and domain knowledge, collectively referred to as *top-down* contributions to attention.

How do bottom-up and top-down contributions to attention interact? Most psychologically and neurobiologically motivated models propose a very similar architecture in which information from bottom-up and top-down sources combines in a *saliency* (or *activation*) map (e.g., Itti et al., 1998; Koch & Ullman, 1985; Mozer, 1991; Wolfe, 1994). The saliency map indicates, for each location in the visual field, the relative importance of that location. Attention is drawn to the most salient locations first.

Figure 1 sketches the basic architecture that incorporates bottom-up and top-down contributions to the saliency map. The visual image is analyzed to extract maps of primitive features such as color and orientation. Associated with each location in a map is a scalar

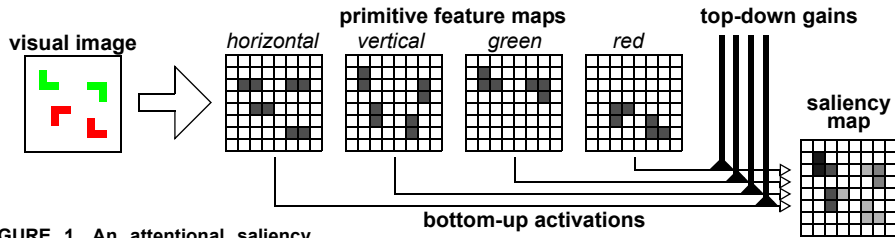

**primitive feature maps**

**top-down gains**

**visual image**

*horizontal*   *vertical*   *green*   *red*

**saliency map**

**bottom-up activations**

FIGURE 1. An attentional saliency map constructed from bottom-up and top-down information

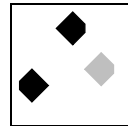

FIGURE 2. Sample display from Experiment 1 of Maljkovic and Nakayama (1994)

response or *activation* indicating the presence of a particular feature. Most models assume that responses are stronger at locations with high local feature contrast, consistent with neurophysiological data, e.g., the response of a red feature detector to a red object is stronger if the object is surrounded by green objects. The saliency map is obtained by taking a sum of bottom-up activations from the feature maps. The bottom-up activations are modulated by a top-down *gain* that specifies the contribution of a particular map to saliency in the current task and environment. Wolfe (1994) describes a heuristic algorithm for determining appropriate gains in a *visual search* task, where the goal is to detect a *target* object among *distractor* objects. Wolfe proposes that maps encoding features that discriminate between target and distractors have higher gains, and to be consistent with the data, he proposes limits on the magnitude of gain modulation and the number of gains that can be modulated. More recently, Wolfe et al. (2003) have been explicit in proposing optimization as a principle for setting gains given the task definition and stimulus environment.

One aspect of optimizing attentional control involves configuring the attentional system to perform a given task; for example, in a visual search task for a red vertical target among green vertical and red horizontal distractors, the task definition should result in a higher gain for red and vertical feature maps than for other feature maps. However, there is a more subtle form of gain modulation, which depends on the statistics of display environments. For example, if green vertical distractors predominate, then red is a better discriminative cue than vertical; and if red horizontal distractors predominate, then vertical is a better discriminative cue than red.

In this paper, we propose a model that encodes statistics of the environment in order to allow for optimization of attentional control to the structure of the environment. Our model is designed to address a key set of behavioral data, which we describe next.

## 1.1 Attentional priming phenomena

Psychological studies involve a sequence of experimental *trials* that begin with a stimulus presentation and end with a response from the human participant. Typically, trial order is randomized, and the context preceding a trial is ignored. However, in *sequential studies*, performance is examined on one trial contingent on the past history of trials. These sequential studies explore how experience influences future performance. Consider a the sequential attentional task of Maljkovic and Nakayama (1994). On each trial, the stimulus display (Figure 2) consists of three notched diamonds, one a singleton in color—either green among red or red among green. The task is to report whether the singleton diamond, referred to as the *target*, is notched on the left or the right. The task is easy because the singleton *pops out*, i.e., the time to locate the singleton does not depend on the number of diamonds in the display. Nonetheless, the response time significantly depends on the sequence of trials leading up to the current trial: If the target is the same color on the cur-

rent trial as on the previous trial, response time is roughly 100 ms faster than if the target is a different color on the current trial. Considering that response times are on the order of 700 ms, this effect, which we term *attentional priming*, is gigantic in the scheme of psychological phenomena.

## 2 ATTENTIONAL CONTROL AS ADAPTATION TO THE STATISTICS OF THE ENVIRONMENT

We interpret the phenomenon of attentional priming via a particular perspective on attentional control, which can be summarized in two bullets.

- The perceptual system dynamically constructs a probabilistic model of the environment based on its past experience.
- Control parameters of the attentional system are tuned so as to optimize performance under the current environmental model.

The primary focus of this paper is the environmental model, but we first discuss the nature of performance optimization.

The role of attention is to make processing of some stimuli more efficient, and consequently, the processing of other stimuli less efficient. For example, if the gain on the red feature map is turned up, processing will be efficient for red items, but competition from red items will reduce the efficiency for green items. Thus, optimal control should tune the system for the most likely states of the world by minimizing an objective function such as:

$$J(\boldsymbol{g}) = \sum_e P(e)RT_{\boldsymbol{g}}(e) \tag{1}$$

where $\boldsymbol{g}$ is a vector of top-down gains, $e$ is an index over environmental states, $P(.)$ is the probability of an environmental state, and $RT_{\boldsymbol{g}}(.)$ is the expected response time—assuming a constant error rate—to the environmental state under gains $\boldsymbol{g}$. Determining the optimal gains is a challenge because every gain setting will result in facilitation of responses to some environmental states but hindrance of responses to other states.

The optimal control problem could be solved via direct reinforcement learning, but the rapidity of human learning makes this possibility unlikely: In a variety of experimental tasks, evidence suggests that adaptation to a new task or environment can occur in just one or two trials (e.g., Rogers & Monsell, 1996). Model-based reinforcement learning is an attractive alternative, because given a model, optimization can occur without further experience in the real world. Although the number of real-world trials necessary to achieve a given level of performance is comparable for direct and model-based reinforcement learning in stationary environments (Kearns & Singh, 1999), naturalistic environments can be viewed as highly nonstationary. In such a situation, the framework we suggest is well motivated: After each experience, the environment model is updated. The updated environmental model is then used to retune the attentional system.

In this paper, we propose a particular model of the environment suitable for visual search tasks. Rather than explicitly modeling the optimization of attentional control by setting gains, we assume that the optimization process will serve to minimize Equation 1. Because any gain adjustment will facilitate performance in some environmental states and hinder performance in others, an optimized control system should obtain faster reaction times for more probable environmental states. This assumption allows us to explain experimental results in a minimal, parsimonious framework.

## 3 MODELING THE ENVIRONMENT

Focusing on the domain of visual search, we characterize the environment in terms of a

probability distribution over configurations of target and distractor features. We distinguish three classes of features: *defining*, *reported,* and *irrelevant*. To explain these terms, consider the task of searching a display of size varying, colored, notched diamonds (Figure 2), with the task of detecting the singleton in color and judging the notch location. Color is the defining feature, notch location is the reported feature, and size is an irrelevant feature. To simplify the exposition, we treat all features as having discrete values, an assumption which is true of the experimental tasks we model. We begin by considering displays containing a single target and a single distractor, and shortly generalize to multidistractor displays.

We use the framework of Bayesian networks to characterize the environment. Each feature of the target and distractor is a discrete random variable, e.g., $T_{color}$ for target color and $D_{notch}$ for the location of the notch on the distractor. The Bayes net encodes the probability distribution over environmental states; in our working example, this distribution is

$$\mathbf{P}(T_{color}, T_{size}, T_{notch}, D_{color}, D_{size}, D_{notch}).$$

The structure of the Bayes net specifies the relationships among the features. The simplest model one could consider would be to treat the features as independent, illustrated in Figure 3a for singleton-color search task. The opposite extreme would be the full joint distribution, which could be represented by a look up table indexed by the six features, or by the cascading Bayes net architecture in Figure 3b. The architecture we propose, which we'll refer to as the *dominance model* (Figure 3c), has an intermediate dependency structure, and expresses the joint distribution as:

$$\mathbf{P}(T_{color})\mathbf{P}(D_{color}|T_{color})\mathbf{P}(T_{size}|T_{color})\mathbf{P}(T_{notch}|T_{color})\mathbf{P}(D_{size}|D_{color})\mathbf{P}(D_{notch}|T_{color}).$$

The structured model is constructed based on three rules.

1. The defining feature of the target is at the root of the tree.

2. The defining feature of the distractor is conditionally dependent on the defining feature of the target. We refer to this rule as *dominance* of the target over the distractor.

3. The reported and irrelevant features of target (distractor) are conditionally dependent on the defining feature of the target (distractor). We refer to this rule as *dominance* of the defining feature over nondefining features.

As we will demonstrate, the dominance model produces a parsimonious account of a wide range of experimental data.

## 3.1  Updating the environment model

The model's parameters are the conditional distributions embodied in the links. In the example of Figure 3c with binary random variables, the model has 11 parameters. However, these parameters are determined by the environment: To be adaptive in nonstationary environments, the model must be updated following each experienced state. We propose a simple exponentially weighted averaging approach. For two variables V and W with observed values $v$ and $w$ on trial $t$, a conditional distribution, $P_t(V = u|W = w) = \delta_{uv}$, is

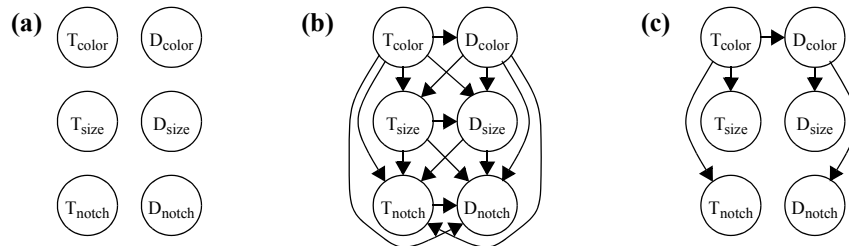

**FIGURE 3. Three models of a visual-search environment with colored, notched, size-varying diamonds. (a) feature-independence model; (b) full-joint model; (c) dominance model.**

defined, where $\delta$ is the Kronecker delta. The distribution representing the environment following trial $t$, denoted $P_t^E$, is then updated as follows:

$$P_t^E(V = u | W = w) = \alpha P_{t-1}^E(V = u | W = w) + (1 - \alpha)P_t(V = u | W = w) \qquad (2)$$

for all $u$, where $\alpha$ is a memory constant. Note that no update is performed for values of W other than $w$. An analogous update is performed for unconditional distributions.

How the model is initialized—i.e., specifying $P_0^E$—is irrelevant, because all experimental tasks that we model, participants begin the experiment with many dozens of practice trials. Data is not collected during practice trials. Consequently, any transient effects of $P_0^E$ do not impact the results. In our simulations, we begin with a uniform distribution for $P_0^E$, and include practice trials as in the human studies.

Thus far, we've assumed a single target and a single distractor. The experiments that we model involve multiple distractors. The simple extension we require to handle multiple distractors is to define a frequentist probability for each distractor feature V, $P_t(V = v | W = w) = C_{vw}/C_w$, where $C_{vw}$ is the count of co-occurrences of feature values $v$ and $w$ among the distractors, and $C_w$ is the count of $w$.

Our model is extremely simple. Given a description of the visual search task and environment, the model has only a single degree of freedom, $\alpha$. In all simulations, we fix $\alpha = 0.75$; however, the choice of $\alpha$ does not qualitatively impact any result.

## 4 SIMULATIONS

In this section, we show that the model can explain a range of data from four different experiments examining attentional priming. All experiments measure response times of participants. On each trial, the model can be used to obtain a probability of the display configuration (the environmental state) on that trial, given the history of trials to that point. Our critical assumption—as motivated earlier—is that response times monotonically decrease with increasing probability, indicating that visual information processing is better configured for more likely environmental states. The particular relationship we assume is that response times are linear in log probability. This assumption yields long response time tails, as are observed in all human studies.

### 4.1 Maljkovic and Nakayama (1994, Experiment 5)

In this experiment, participants were asked to search for a singleton in color in a display of three red or green diamonds. Each diamond was notched on either the left or right side, and the task was to report the side of the notch on the color singleton. The well-practiced participants made very few errors. Reaction time (RT) was examined as a function of whether the target on a given trial is the same or different color as the target on trial $n$ steps back or ahead. Figure 4 shows the results, with the human RTs in the left panel and the simulation log probabilities in the right panel. The horizontal axis represents $n$. Both graphs show the same outcome: repetition of target color facilitates performance. This influence lasts only for a half dozen trials, with an exponentially decreasing influence further into the past. In the model, this decreasing influence is due to the exponential decay of recent history (Equation 2). Figure 4 also shows that—as expected—the future has no influence on the current trial.

### 4.2 Maljkovic and Nakayama (1994, Experiment 8)

In the previous experiment, it is impossible to determine whether facilitation is due to repetition of the target's color or the distractor's color, because the display contains only two colors, and therefore repetition of target color implies repetition of distractor color. To unconfound these two potential factors, an experiment like the previous one was con-

ducted using four distinct colors, allowing one to examine the effect of repeating the target color while varying the distractor color, and vice versa. The sequence of trials was composed of subsequences of up-to-six consecutive trials with either the target or distractor color held constant while the other color was varied trial to trial. Following each subsequence, both target and distractors were changed. Figure 5 shows that for both humans and the simulation, performance improves toward an asymptote as the number of target and distractor repetitions increases; in the model, the asymptote is due to the probability of the repeated color in the environment model approaching 1.0. The performance improvement is greater for target than distractor repetition; in the model, this difference is due to the dominance of the defining feature of the target over the defining feature of the distractor.

### 4.3 Huang, Holcombe, and Pashler (2004, Experiment 1)

Huang et al. (2004) and Hillstrom (2000) conducted studies to determine whether repetitions of one feature facilitate performance independently of repetitions of another feature. In the Huang et al. study, participants searched for a singleton in size in a display consisting of lines that were short and long, slanted left or right, and colored white or black. The reported feature was target slant. Slant, size, and color were uncorrelated. Huang et al. discovered that repeating an irrelevant feature (color or orientation) facilitated performance, but only when the defining feature (size) was repeated. As shown in Figure 6, the model replicates human performance, due to the dominance of the defining feature over the reported and irrelevant features.

### 4.4 Wolfe, Butcher, Lee, and Hyde (2003, Experiment 1)

In an empirical tour-de-force, Wolfe et al. (2003) explored singleton search over a range of environments. The task is to detect the presence or absence of a singleton in displays con-

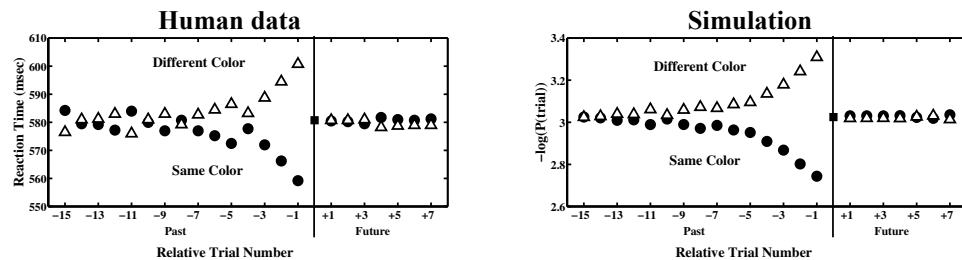

**FIGURE 4. Experiment 5 of Maljkovic and Nakayama (1994): performance on a given trial conditional on the color of the target on a previous or subsequent trial. Human data is from subject KN.**

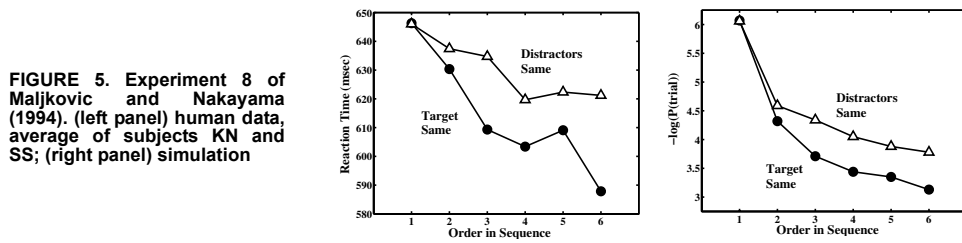

**FIGURE 5. Experiment 8 of Maljkovic and Nakayama (1994). (left panel) human data, average of subjects KN and SS; (right panel) simulation**

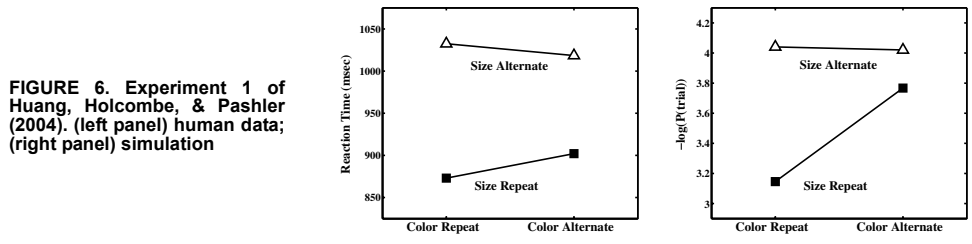

**FIGURE 6. Experiment 1 of Huang, Holcombe, & Pashler (2004). (left panel) human data; (right panel) simulation**

sisting of colored (red or green), oriented (horizontal or vertical) lines. Target-absent trials were used primarily to ensure participants were searching the display. The experiment examined seven experimental conditions, which varied in the amount of uncertainty as to the target identity. The essential conditions, from least to most uncertainty, are: *blocked* (e.g., target always red vertical among green horizontals), *mixed feature* (e.g., target always a color singleton), *mixed dimension* (e.g., target either red or vertical), and *fully mixed* (target could be red, green, vertical, or horizontal). With this design, one can ascertain how uncertainty in the environment and in the target definition influence task difficulty. Because the defining feature in this experiment could be either color or orientation, we modeled the environment with two Bayes nets—one color dominant and one orientation dominant—and performed model averaging. A comparison of Figures 7a and 7b show a correspondence between human RTs and model predictions. Less uncertainty in the environment leads to more efficient performance. One interesting result from the model is its prediction that the mixed-feature condition is easier than the fully-mixed condition; that is, search is more efficient when the *dimension* (i.e., color vs. orientation) of the singleton is known, even though the model has no abstract representation of feature dimensions, only feature values.

## 4.5 Optimal adaptation constant

In all simulations so far, we fixed the memory constant. From the human data, it is clear that memory for recent experience is relatively short lived, on the order of a half dozen trials (e.g., left panel of Figure 4). In this section we provide a rational argument for the short duration of memory in attentional control.

Figure 7c shows mean negative log probability in each condition of the Wolfe et al. (2003) experiment, as a function of $\alpha$. To assess these probabilities, for each experimental condition, the model was initialized so that all of the conditional distributions were uniform, and then a block of trials was run. Log probability for all trials in the block was averaged. The negative log probability (y axis of the Figure) is a measure of the model's *misprediction* of the next trial in the sequence.

For complex environments, such as the fully-mixed condition, a small memory constant is detrimental: With rapid memory decay, the effective history of trials is a high-variance sample of the distribution of environmental states. For simple environments, a large memory constant is detrimental: With slow memory decay, the model does not transition quickly from the initial environmental model to one that reflects the statistics of a new environment. Thus, the memory constant is constrained by being large enough that the environment model can hold on to sufficient history to represent complex environments, and by being small enough that the model adapts quickly to novel environments. If the conditions in Wolfe et al. give some indication of the range of naturalistic environments an agent encounters, we have a rational account of why attentional priming is so short lived. Whether priming lasts 2 trials or 20, the surprising empirical result is that it does not last 200 or 2000 trials. Our rational argument provides a rough insight into this finding.

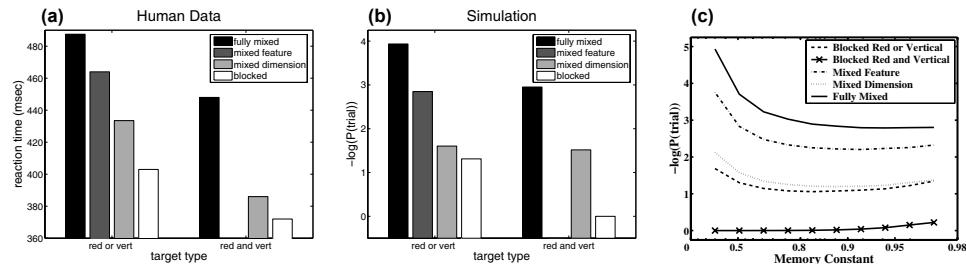

FIGURE 7. (a) Human data for Wolfe et al. (2003), Experiment 1; (b) simulation; (c) *misprediction* of model (i.e., lower y value = better) as a function of $\alpha$ for five experimental condition

# 5 DISCUSSION

The psychological literature contains two opposing accounts of attentional priming and its relation to attentional control. Huang et al. (2004) and Hillstrom (2000) propose an *episodic* account in which a distinct memory trace—representing the complete configuration of features in the display—is laid down for each trial, and priming depends on configural similarity of the current trial to previous trials. Alternatively, Maljkovic and Nakayama (1994) and Wolfe et al. (2003) propose a *feature-strengthening* account in which detection of a feature on one trial increases its ability to attract attention on subsequent trials, and priming is proportional to the number of overlapping features from one trial to the next. The episodic account corresponds roughly to the full joint model (Figure 3b), and the feature-strengthening account corresponds roughly to the independence model (Figure 3a). Neither account is adequate to explain the range of data we presented. However, an intermediate account, the dominance model (Figure 3c), is not only sufficient, but it offers a parsimonious, rational explanation. Beyond the model's basic assumptions, it has only one free parameter, and can explain results from diverse experimental paradigms.

The model makes a further theoretical contribution. Wolfe et al. distinguish the environments in their experiment in terms of the amount of top-down control available, implying that different mechanisms might be operating in different environments. However, in our account, top-down control is not some substance distributed in different amounts depending on the nature of the environment. Our account treats all environments uniformly, relying on attentional control to adapt to the environment at hand.

We conclude with two limitations of the present work. First, our account presumes a particular network architecture, instead of a more elegant Bayesian approach that specifies priors over architectures, and performs automatic model selection via the sequence of trials. We did explore such a Bayesian approach, but it was unable to explain the data. Second, at least one finding in the literature is problematic for the model. Hillstrom (2000) occasionally finds that RTs slow when an irrelevant target feature is repeated but the defining target feature is not. However, because this effect is observed only in some experiments, it is likely that *any* model would require elaboration to explain the variability.

## ACKNOWLEDGEMENTS

We thank Jeremy Wolfe for providing the raw data from his experiment for reanalysis. This research was funded by NSF BCS Award 0339103.

## REFERENCES

Huang, L, Holcombe, A. O., & Pashler, H. (2004). Repetition priming in visual search: Episodic retrieval, not feature priming. *Memory & Cognition, 32*, 12–20.

Hillstrom, A. P. (2000). Repetition effects in visual search. *Perception & Psychophysics*, **62**, 800-817.

Itti, L., Koch, C., & Niebur, E. (1998). A model of saliency-based visual attention for rapid scene analysis. *IEEE Trans. Pattern Analysis & Machine Intelligence*, **20,** 1254–1259.

Kearns, M., & Singh, S. (1999). Finite-sample convergence rates for Q-learning and indirect algorithms. In *Advances in Neural Information Processing Systems 11* (pp. 996–1002). Cambridge, MA: MIT Press.

Koch, C. and Ullman, S. (1985). Shifts in selective visual attention: towards the underlying neural circuitry. *Human Neurobiology*, **4**, 219–227.

Maljkovic, V., & Nakayama, K. (1994). Priming of pop-out: I. Role of features. *Mem. & Cognition*, **22**, 657-672.

Mozer, M. C. (1991). The perception of multiple objects: A connectionist approach. Cambridge, MA: MIT Press.

Rogers, R. D., & Monsell, S. (1995). The cost of a predictable switch between simple cognitive tasks. *Journal of Experimental Psychology: General, 124*, 207–231.

Wolfe, J.M. (1994). Guided Search 2.0: A Revised Model of Visual Search. *Psych. Bull. & Rev.*, **1**, 202–238.

Wolfe, J. S., Butcher, S. J., Lee, C., & Hyde, M. (2003). Changing your mind: on the contributions of top-down and bottom-up guidance in visual search for feature singletons. *Journal of Exptl. Psychology: Human Perception & Performance*, **29**, 483-502.
